# Multiclass Learning by Probabilistic Embeddings

**Ofer Dekel** and **Yoram Singer**
School of Computer Science & Engineering
The Hebrew University, Jerusalem 91904, Israel
{oferd,singer}@cs.huji.ac.il

## Abstract

We describe a new algorithmic framework for learning multiclass categorization problems. In this framework a multiclass predictor is composed of a pair of embeddings that map both instances and labels into a common space. In this space each instance is assigned the label it is nearest to. We outline and analyze an algorithm, termed Bunching, for learning the pair of embeddings from labeled data. A key construction in the analysis of the algorithm is the notion of probabilistic output codes, a generalization of error correcting output codes (ECOC). Furthermore, the method of multiclass categorization using ECOC is shown to be an instance of Bunching. We demonstrate the advantage of Bunching over ECOC by comparing their performance on numerous categorization problems.

## 1 Introduction

The focus of this paper is supervised learning from multiclass data. In multiclass problems the goal is to learn a classifier that accurately assigns labels to instances where the set of labels is of finite cardinality and contains more than two elements. Many machine learning applications employ a multiclass categorization stage. Notable examples are document classification, spoken dialog categorization, optical character recognition (OCR), and part-of-speech tagging. Dietterich and Bakiri [6] proposed a technique based on error correcting output coding (ECOC) as a means of reducing a multiclass classification problem to several binary classification problems and then solving each binary problem individually to obtain a multiclass classifier. More recent work of Allwein et al. [1] provided analysis of the empirical and generalization errors of ECOC-based classifiers. In the above papers, as well as in most previous work on ECOC, learning the set of binary classifiers and selecting a particular error correcting code are done independently. An exception is a method based on continuous relaxation of the code [3] in which the code matrix is *post-processed once* based on the learned binary classifiers.

The inherent decoupling of the learning process from the class representation problem employed by ECOC is both a blessing and a curse. On one hand it offers great flexibility and modularity, on the other hand, the resulting binary learning problems might be unnatural and therefore potentially difficult. We instead describe and analyze an approach that ties the learning problem with the class representation problem. The approach we take perceives the set of binary classifiers as an embedding of the instance space and the code matrix as an embedding of the label set into a common space. In this common space each instance is assigned the label from which it's divergence is smallest. To construct these embeddings, we introduce the notion of probabilistic output codes. We then describe an algorithm that constructs the label and instance embeddings such that the resulting classifier achieves a small empirical error. The result is a paradigm that includes ECOC as a special case.

The algorithm we describe, termed Bunching, alternates between two steps. One step improves the embedding of the instance space into the common space while keeping the embedding of the label set fixed. This step is analogous to the learning stage of the ECOC technique, where a set of binary classifiers are learned with respect to a predefined code. The second step complements the first by updating the label embedding while keeping the instance embedding fixed. The two alternating steps resemble the steps performed by the EM algorithm [5] and by Alternating Minimization [4]. The techniques we use in the design and analysis of the Bunching algorithm also build on recent results in classification learning using Bregman divergences [8, 2].

The paper is organized as follows. In the next section we give a formal description of the multiclass learning problem and of our classification setting. In Sec. 3 we give an alternative view of ECOC which naturally leads to the definition of probabilistic output codes presented in Sec. 4. In Sec. 5 we cast our learning problem as a minimization problem of a continuous objective function and in Sec. 6 we present the Bunching algorithm. We describe experimental results that demonstrate the merits of our approach in Sec. 7 and conclude in Sec. 8.

## 2 Problem Setting

Let $\mathcal{X}$ be a domain of instance encodings from $\mathbb{R}^m$ and let $\mathcal{Y}$ be a set of $r$ labels that can be assigned to each instance from $\mathcal{X}$. Given a training set of instance-label pairs $S = (x_j, y_j)_{j=1}^n$ such that each $x_j$ is in $\mathcal{X}$ and each $y_j$ is in $\mathcal{Y}$, we are faced with the problem of learning a classification function that predicts the labels of instances from $\mathcal{X}$. This problem is often referred to as multiclass learning. In other multiclass problem settings it is common to encode the set $\mathcal{Y}$ as a prefix of the integers $\{1, \ldots, r\}$, however in our setting it will prove useful to assume that the labels are encoded as the set of $r$ standard unit vectors in $\mathbb{R}^r$. That is, the $i$'th label in $\mathcal{Y}$ is encoded by the vector whose $i$'th component is set to 1, and all of its other components are set to 0.

The classification functions we study in this paper are composed of a pair of embeddings from the spaces $\mathcal{X}$ and $\mathcal{Y}$ into a common space $\mathcal{Z}$, and a measure of divergence between vectors in $\mathcal{Z}$. That is, given an instance $x \in \mathcal{X}$, we embed it into $\mathcal{Z}$ along with all of the label vectors in $\mathcal{Y}$ and predict the label that $x$ is closest to in $\mathcal{Z}$. The measure of distance between vectors in $\mathcal{Z}$ builds upon the definitions given below:

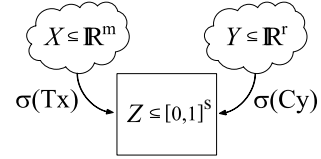

Figure 1: An illustration of the embedding model used.

The *logistic transformation* $\sigma : \mathbb{R}^s \to (0, 1)^s$ is defined

$$\forall k = 1, ..., s \quad \sigma_k(\omega) = (1 + e^{-\omega_k})^{-1}$$

The *entropy* of a multivariate Bernoulli random variable with parameter $p \in [0, 1]^s$ is

$$H[p] = -\sum_{k=1}^{s} \left[ p_k \log(p_k) + (1 - p_k) \log(1 - p_k) \right] .$$

The *Kullback-Leibler (KL) divergence* between a pair of multivariate Bernoulli random variables with respective parameters $p, q \in [0, 1]^s$ is

$$D[p \parallel q] = \sum_{k=1}^{s} \left[ p_k \log\left(\frac{p_k}{q_k}\right) + (1 - p_k) \log\left(\frac{1 - p_k}{1 - q_k}\right) \right] . \tag{1}$$

Returning to our method of classification, let $s$ be some positive integer and let $\mathcal{Z}$ denote the space $[0, 1]^s$. Given any two linear mappings $T : \mathbb{R}^m \to \mathbb{R}^s$ and $C : \mathbb{R}^r \to \mathbb{R}^s$, where $T$ is given as a matrix in $\mathbb{R}^{s \times m}$ and $C$ as a matrix in $\mathbb{R}^{s \times r}$, instances from $\mathcal{X}$ are embedded into $\mathcal{Z}$ by $\sigma(Tx)$ and labels from $\mathcal{Y}$ are embedded into $\mathcal{Z}$ by $\sigma(Cy)$. An illustration of the two embeddings is given in Fig. 1.

We define the divergence between any two points $z_1, z_2 \in \mathcal{Z}$ as the sum of the KL-divergence between them and the entropy of $z_1$, $D[z_1 \parallel z_2] + H[z_1]$. We now define the loss $\ell$ of each instance-label pair as the divergence of their respective images,

$$\ell(x, y|C, T) = D[\sigma(Cy) \parallel \sigma(Tx)] + H[\sigma(Cy)] . \tag{2}$$

This loss is clearly non-negative and can be zero iff $x$ and $y$ are embedded to the same point in $\mathcal{Z}$ and the entropy of this point is zero. $\ell$ is our means of classifying new instances: given a new instance we predict its label to be $\hat{y}$ if

$$\hat{y} = \operatorname*{argmin}_{y \in \mathcal{Y}} \ell(x, y|C, T) . \tag{3}$$

For brevity, we restrict ourselves to the case where only a single label attains the minimum loss, and our classifier is thus always well defined. We point out that our analysis is still valid when this constraint is relaxed. We name the loss over the entire training set $S$ the empirical loss and use the notation

$$\mathcal{L}(S|C, T) = \sum_{(x,y) \in S} \ell(x, y|C, T) . \tag{4}$$

Our goal is to learn a good multiclass prediction function by finding a pair $(C, T)$ that attains a small empirical loss. As we show in the sequel, the rationale behind this choice of empirical loss lies in the fact that it bounds the (discrete) empirical classification error attained by the classification function.

## 3 An Alternative View of Error Correcting Output Codes

The technique of ECOC uses error correcting codes to reduce an $r$-class classification problem to multiple binary problems. Each binary problem is then learned independently via an external binary learning algorithm and the learned binary classifiers are combined into one $r$-class classifier. We begin by giving a brief overview of ECOC for the case where the binary learning algorithm used is a logistic regressor.

A binary output code $C$ is a matrix in $\{0,1\}^{s \times r}$ where each of $C$'s columns is an $s$-bit code word that corresponds to a label in $\mathcal{Y}$. Recall that the set of labels $\mathcal{Y}$ is assumed to be the standard unit vectors in $\mathbb{R}^r$. Therefore, the code word corresponding to the label $y$ is simply the product of the matrix $C$ and the vector $y$, $Cy$. The distance $\rho$ of a code $C$ is defined as the minimal Hamming distance between any two code words, formally

$$\rho(C) = \min_{i \neq j} \sum_{k=1}^{s} C_{k,i}(1 - C_{k,j}) + C_{k,j}(1 - C_{k,i}) .$$

For any $k \in \{1, \ldots, s\}$, the $k$'th row of $C$, denoted henceforth by $C_k$, defines a partition of the set of labels $\mathcal{Y}$ into two disjoint subsets: the first subset constitutes labels for which $C_k \cdot y = 0$ (i.e., the set of labels in $\mathcal{Y}$ which are mapped according to $C_k$ to the binary label 0) and the labels for which $C_k \cdot y = 1$. Thus, each $C_k$ induces a binary classification problem from the original multiclass problem. Formally, we construct for each $k$ a binary-labeled sample $S_k = \{(x_j, C_k \cdot y_j)\}_{j=1}^{n}$ and for each $S_k$ we learn a binary classification function $T_k : \mathcal{X} \to \mathbb{R}$ using a logistic regression algorithm. That is, for each original instance $x_j$ and induced binary label $C_k \cdot y_j$ we posit a logistic model that estimates the conditional probability that $C_k \cdot y_j$ equals 1 given $x_j$,

$$Pr[C_k \cdot y_j = 1| \ x_j \ ; \ T_k] = \sigma(T_k \cdot x_j) . \tag{5}$$

Given a *predefined* code matrix $C$ the learning task at hand is to find $T_k^\star$ that maximizes the log-likelihood of the labelling given in $S_k$,

$$T_k^\star = \operatorname*{argmax}_{T_k \in \mathbb{R}^m} \sum_{j=1}^{n} \log(Pr[C_k \cdot y_j \mid x_j \ ; \ T_k]) . \tag{6}$$

Defining $0 \log 0 = 0$, we can use the logistic estimate in Eq. (5) and the KL-divergence from Eq. (1) to rewrite Eq. (6) as follows

$$T_k^\star = \underset{T_k \in \mathbb{R}^m}{\operatorname{argmin}} \sum_{j=1}^{n} D[C_k \cdot y_j \parallel \sigma(T_k \cdot x_j)] \, .$$

In words, a good set of binary predictors is found by minimizing the sample-averaged KL-divergence between the binary vectors induced by $C$ and the logistic estimates induced by $T_1, \ldots, T_s$. Let $T^\star$ be the matrix in $\mathbb{R}^{s \times m}$ constructed by the concatenation of the row vectors $\{T_k^\star\}_{k=1}^s$. For any instance $x \in \mathcal{X}$, $\sigma(T^\star x)$ is a vector of probability estimates that the label of $x$ is 1 for each of the $s$ induced binary problems. We can summarize the learning task defined by the code $C$ as the task of finding a matrix $T^\star$ such that

$$T^\star = \underset{T \in \mathbb{R}^{s \times m}}{\operatorname{argmin}} \sum_{j=1}^{n} D[Cy_j \parallel \sigma(Tx_j)] \, .$$

Given a code matrix $C$ and a transformation $T$ we classify a new instance as follows,

$$\hat{y} = \underset{y \in \mathcal{Y}}{\operatorname{argmin}} \, D[Cy \parallel \sigma(Tx)] \, . \tag{7}$$

A classification error occurs if the predicted label $\hat{y}$ is different from the correct label $y$. Building on Thm. 1 from Allwein et al. [1] it is straightforward to show that the empirical classification error ($\hat{y} \neq y$) is bounded above by the empirical KL-divergence between the correct code word $Cy$ and the estimated probabilities $\sigma(Tx)$ divided by the code distance,

$$|\{\hat{y}_j \neq y_j\}_{j=1}^n| \leq \frac{\sum_{j=1}^{n} D[Cy_j \parallel \sigma(Tx_j)]}{\rho(C)} \, . \tag{8}$$

This bound is a special case of the bound given below in Thm. 1 for general probabilistic output codes. We therefore defer the discussion on this bound to the following section.

## 4    Probabilistic Output Codes

We now describe a relaxation of binary output codes by defining the notion of probabilistic output codes. We give a bound on the empirical error attained by a classifier that uses probabilistic output codes which generalizes the bound in Eq. (8). The rationale for our construction is that the discrete nature of ECOC can potentially induce difficult binary classification problems. In contrast, probabilistic codes induce real-valued problems that may be easier to learn.

Analogous to discrete codes, A probabilistic output code $C$ is a matrix in $\mathbb{R}^{s \times r}$ used in conjunction with the logistic transformation to produce a set of $r$ probability vectors that correspond to the $r$ labels in $\mathcal{Y}$. Namely, $C$ maps each label $y \in \mathcal{Y}$ to the probabilistic code word $\sigma(Cy) \in [0,1]^s$. As before, we assume that $\mathcal{Y}$ is the set of $r$ standard unit vectors in $\{0,1\}^r$ and therefore each probabilistic code word is the image of one of $C$'s columns under the logistic transformation. The natural extension of code distance to probabilistic codes is achieved by replacing Hamming distance with expected Hamming distance. If for each $y \in \mathcal{Y}$ and $k \in \{1, \ldots, s\}$ we view the $k$'th component of the code word that corresponds to $y$ as a Bernoulli random variable with parameter $p = \sigma_k(Cy)$ then the expected Hamming distance between the code word for classes $i$ and $j$ is,

$$\sum_{k=1}^{s} \sigma_k(Cy_i)(1 - \sigma_k(Cy_j)) + \sigma_k(Cy_j)(1 - \sigma_k(Cy_i)) \, .$$

Analogous to discrete codes we define the distance $\rho$ of a code $C$ as the minimum expected Hamming distance between all pairs of code words in $C$, that is,

$$\rho(C) = \min_{i \neq j} \sum_{k=1}^{s} \sigma_k(Cy_i)(1 - \sigma_k(Cy_j)) + \sigma_k(Cy_j)(1 - \sigma_k(Cy_i)) \ .$$

Put another way, we have relaxed the definition of code words from deterministic vectors to multivariate Bernoulli random variables. The matrix $C$ now defines the distributions of these random variables. When $C$'s entries are all $\pm\infty$ then the logistic transformation of $C$'s entries defines a deterministic code and the two definitions of $\rho$ coincide.

Given a probabilistic code matrix $C \in \mathbb{R}^{s \times r}$ and a transformation $T \in \mathbb{R}^{s \times m}$ we associate a loss $\ell(x, y | C, T)$ with each instance-label pair $(x, y)$ using Eq. (2) and we measure the empirical loss over the entire training set $S$ as defined in Eq. (4). We classify new instances by finding the label $\hat{y}$ that attains the smallest loss as defined in Eq. (3). This construction is equivalent to the classification method discussed in Sec. 2 that employs embeddings except that instead of viewing $C$ and $T$ as abstract embeddings $C$ is interpreted as a probabilistic output code and the rows of $T$ are viewed as binary classifiers.

Note that when all of the entries of $C$ are $\pm\infty$ then the classification rule from Eq. (3) is reduced to the classification rule for ECOC from Eq. (7) since the entropy of $\sigma(Cy)$ is zero for all $y$. We now give a theorem that builds on our construction of probabilistic output codes and relates the classification rule from Eq. (3) with the empirical loss defined by Eq. (4). As noted before, the theorem generalizes the bound given in Eq. (8).

**Theorem 1** *Let $\mathcal{Y}$ be a set of $r$ vectors in $\mathbb{R}^r$. Let $C \in \mathbb{R}^{s \times r}$ be a probabilistic output code with distance $\rho(C)$ and let $T \in \mathbb{R}^{s \times m}$ be a transformation matrix. Given a sample $S = \{(x_j, y_j)\}_{i=j}^{n}$ of instance-label pairs where $x_j \in \mathcal{X}$ and $y_j \in \mathcal{Y}$, denote by $\mathcal{L}$ the loss on $S$ with respect to $C$ and $T$ as given by Eq. (4) and denote by $\hat{y}_j$ the predicted label of $x_j$ according to the classification rule given in Eq. (3). Then,*

$$|\{\hat{y}_j \neq y_j\}_{j=1}^{n}| \leq \frac{\mathcal{L}(S|C, T)}{\rho(C)} \ .$$

The proof of the theorem is omitted due to the lack of space.

## 5   The Learning Problem

We now discuss how our formalism of probabilistic output codes via embeddings and the accompanying Thm. 1 lead to a learning paradigm in which both $T$ and $C$ are found concurrently. Thm. 1 implies that the empirical error over $S$ can be reduced by minimizing the empirical loss over $S$ while maintaining a large distance $\rho(C)$. A naive modification of $C$ so as to minimize the loss may result in a probabilistic code whose distance is undesirably small. Therefore, we assume that we are initially provided with a fixed reference matrix $C_0 \in \mathbb{R}^{s \times r}$ that is known to have a large code distance. We now require that the *learned* matrix $C$ remain relatively close to $C_0$ (in a sense defined shortly) throughout the learning procedure. Rather than requiring that $C$ attain a fixed distance to $C_0$ we add a penalty proportional to the distance between $C$ and $C_0$ to the loss defined in Eq. (4). This penalty on $C$ can be viewed as a form of regularization (see for instance [10]). Similar paradigms have been used extensively in the pioneering work of Warmuth and his colleagues on online learning (see for instance [7] and the references therein) and more recently for incorporating prior knowledge into boosting [11]. The regularization factor we employ is the KL-divergence between the images of $C$ and $C_0$ under the logistic transformation,

$$\mathcal{R}(S|C, C_0) = \sum_{j=1}^{n} D[\sigma(Cy_j) \| \sigma(C_0 y_j)] \ .$$

The influence of this penalty term is controlled by a parameter $\alpha \in [0, \infty]$. The resulting objective function that we attempt to minimize is

$$\mathcal{O}(S|C, T) = \mathcal{L}(S|C, T) + \alpha \mathcal{R}(S|C, C_0) \tag{9}$$

where $\alpha$ and $C_0$ are fixed parameters. The goal of learning boils down to finding a pair $(C^\star, T^\star)$ that minimizes the objective function defined in Eq. (9). We would like to note that this objective function is not convex due to the concave entropic term in the definition of $\ell$. Therefore, the learning procedure described in the sequel converges to a local minimum or a saddle point of $\mathcal{O}$.

## 6  The Learning Algorithm

The goal of the learning algorithm is to find $C$ and $T$ that minimize the objective function defined above. The algorithm alternates between two complementing steps for decreasing the objective function. The first step, called IMPROVE-T, improves $T$ leaving $C$ unchanged, and the second step, called IMPROVE-C, finds the optimal matrix $C$ for any given matrix $T$. The algorithm is provided with initial matrices $C_0$ and $T_0$, where $C_0$ is assumed to have a large code distance $\rho$. The IMPROVE-T step makes the assumption that all of the instances in $S$ satisfy the constraints $\sum_{i=1}^{m} x_i \leq 1$ and for all $i \in \{1, 2, ..., m\}$, $0 \leq x_i$. Any finite training set can be easily shifted and scaled to conform with these constraints and therefore they do not impose any real limitation. In addition, the IMPROVE-C step is presented for the case where $\mathcal{Y}$ is the set of standard unit vectors in $\mathbb{R}^r$.

---

BUNCH $\left( S, \;\; \alpha \in \mathbb{R}^+, \;\; C_0 \in \mathbb{R}^{s \times r}, \;\; T_0 \in \mathbb{R}^{s \times m} \right)$
&emsp;**For** $t = 1, 2, ...$
&emsp;&emsp;$T_t = $ IMPROVE-T $(S, C_{t-1}, T_{t-1})$
&emsp;&emsp;$C_t = $ IMPROVE-C $(S, \alpha, T_t, C_0)$

IMPROVE-T $(S, \;\; C, \;\; T)$
&emsp;**For** $k = 1, 2, ..., s \;\;$ and $\;\; i = 1, 2, ..., m$

$$W_{k,i}^+ = \sum_{(x,y) \in S} \sigma(C_k y) \, \sigma(-T_k x) \, x_i$$

$$W_{k,i}^- = \sum_{(x,y) \in S} \sigma(-C_k y) \, \sigma(T_k x) \, x_i$$

$$\Theta_{k,i} = \frac{1}{2} \ln \left( \frac{W_{k,i}^+}{W_{k,i}^-} \right)$$

&emsp;**Return** $T + \Theta$

IMPROVE-C $(S, \;\; \alpha, \;\; T, \;\; C_0)$
&emsp;**For each** $y \in \mathcal{Y}$
&emsp;&emsp;$S_y = \{(x, \bar{y}) \in S \;\; : \;\; \bar{y} = y\}$

$$C^{(y)} = C_0^{(y)} + \frac{1}{\alpha |S_y|} \sum_{x \in S_y} Tx$$

&emsp;**Return** $C = \left( C^{(1)}, \ldots, C^{(r)} \right)$

Figure 2: The Bunching Algorithm.

---

Since the regularization factor $\mathcal{R}$ is independent of $T$ we can restrict our description and analysis of the IMPROVE-T step to considering only the loss term $\mathcal{L}$ of the objective function $\mathcal{O}$. The IMPROVE-T step receives the current matrices $C$ and $T$ as input and calculates a matrix $\Theta$ that is used for updating the current $T$ additively. Denoting the iteration index by $t$, the update is of the form $T_{t+1} = T_t + \Theta$. The next theorem states that updating $T$ by the IMPROVE-T step decreases the loss or otherwise $T$ remains unchanged and is globally optimal with respect to $C$. Again, the proof is omitted due to space constraints.

**Theorem 2** *Given matrices $C \in \mathbb{R}^{s \times r}$ and $T \in \mathbb{R}^{s \times m}$, let $W_{k,i}^+$, $W_{k,i}^-$ and $\Theta$ be as defined in the* IMPROVE-T *step of Fig. 2. Then, the decrease in the loss $\mathcal{L}$ is bounded below by,*

$$\sum_{k=1}^{s} \sum_{i=1}^{m} \left( \sqrt{W_{k,i}^+} - \sqrt{W_{k,i}^-} \right)^2 \leq \mathcal{L}(S|C,T) - \mathcal{L}(S|C, T + \Theta) \, .$$

Based on the theorem above we can derive the following corollary

**Corollary 1** *If $\Theta$ is generated by a call to* IMPROVE-T *and $\mathcal{L}(S|C, T+\Theta) = \mathcal{L}(S|C,T)$ then $\Theta$ is the zero matrix and $T$ is globally optimal with respect to $C$.*

In the IMPROVE-C step we fix the current matrix $T$ and find a code matrix $C$ that globally minimizes the objective function. According to the discussion above, the matrix $C$ defines

an embedding of the label vectors from $\mathcal{Y}$ into $\mathcal{Z}$ and the images of this embedding constitute the classification rule. For each $y \in Y$ denote its image under $C$ and the logistic transformation by $p_y = \sigma(Cy)$ and let $S_y$ be the subset of $S$ that is labeled $y$. Note that the objective function can be decomposed into $r$ separate summands according to $y$,

$$\mathcal{O}(S|C,T) = \sum_{y \in \mathcal{Y}} \mathcal{O}(S_y|C,T) \ ,$$

where

$$\mathcal{O}(S_y|C,T) = \sum_{(x,y) \in S_y} D[p_y \parallel \sigma(Tx)] + H[p_y] + \alpha D[p_y \parallel \sigma(C_0 y_0)] \ .$$

We can therefore find for each $y \in \mathcal{Y}$ the vector $p_y$ that minimizes $\mathcal{O}(S_y)$ independently and then reconstruct the code matrix $C$ that achieves these values. It is straightforward to show that $\mathcal{O}(S_y)$ is convex in $p_y$, and our task is reduced to finding it's stationary point. We examine the derivative of $\mathcal{O}(S_y)$ with respect to $p_{y,k}$ and get,

$$\frac{\partial \mathcal{O}_y(S_y)}{\partial p_{y,k}} = \sum_{(x,y) \in S_y} -\log\left(\frac{\sigma(T_k \cdot x)}{1 - \sigma(T_k \cdot x)}\right) - \alpha|S_y|\left(C_{0,k} \cdot y + \log\left(\frac{p_{y,k}}{1 - p_{y,k}}\right)\right) \ .$$

We now plug $p_y = \sigma(Cy)$ into the equation above and evaluate it at zero to get that,

$$Cy = C_0 y + \frac{1}{\alpha|S_y|} \sum_{(x,y) \in S_y} Tx \ .$$

Since $\mathcal{Y}$ was assumed to be the set of standard unit vectors, $Cy$ is a column of $C$ and the above is simply a column wise assignment of $C$.

We have shown that each call to IMPROVE-T followed by IMPROVE-C decreases the objective function until convergence to a pair $(C^\star, T^\star)$ such that $C^\star$ is optimal given $T^\star$ and $T^\star$ is optimal given $C^\star$. Therefore $\mathcal{O}(S|C^\star, T^\star)$ is either a minimum or a saddle point.

## 7 Experiments

To assess the merits of Bunching we compared it to a standard ECOC-based algorithm on numerous multiclass problems. For the ECOC-based algorithm we used a logistic regressor as the binary learning algorithm, trained using the parallel update described in [2]. The two approaches share the same form of classifiers (logistic regressors) and differ solely in the coding matrix they employ: while ECOC uses a fixed code matrix Bunching adapts its code matrix during the learning process.

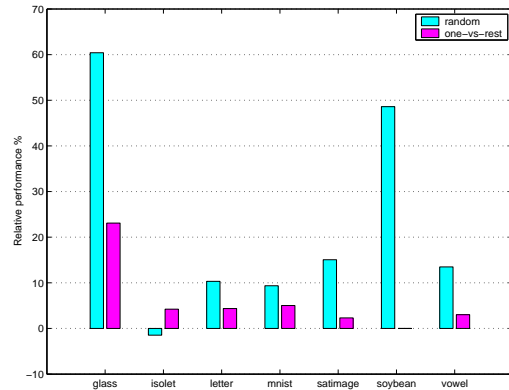

Figure 3: The relative performance of Bunching compared to ECOC on various datasets.

We selected the following multiclass datasets: `glass`, `isolet`, `letter`, `satimage`, `soybean` and `vowel` from the UCI repository (www.ics.uci.edu/~mlearn/MLRepository.html) and the `mnist` dataset available from LeCun's homepage (yann.lecun.com/exdb/mnist/index.html). The only dataset not supplied with a test set is `glass` for which we use 5-fold cross validation.

For each dataset, we compare the test error rate attained by the ECOC classifier and the Bunching classifier. We conducted the experiments for two families of code matrices. The

first family corresponds to the one-vs-rest approach in which each class is trained against the rest of the classes and the corresponding code is a matrix whose logistic transformation is simply the identity matrix. The second family is the set of random code matrices with $r \log_2 r$ rows where $r$ is the number of different labels. These matrices are used as $C_0$ for Bunching and as the fixed code for ECOC. Throughout all of the experiments with Bunching, we set the regularization parameter $\alpha$ to 1.

A summary of the results is depicted in Fig. 3. The height of each bar is proportional to $(e_E - e_B)/e_E$ where $e_E$ is the test error attained by the ECOC classifier and $e_B$ is the test error attained by the Bunching classifier. As shown in the figure, for almost all of the experiments conducted Bunching outperforms standard ECOC. The improvement is more significant when using random code matrices. This can be explained by the fact that random code matrices tend to induce unnatural and rather difficult binary partitions of the set of labels. Since Bunching modifies the code matrix $C$ along its run, it can relax difficult binary problems. This suggests that Bunching can improve the classification accuracy in problems where, for instance, the one-vs-rest approach fails to give good results or when there is a need to add error correction properties to the code matrix.

## 8 A Brief Discussion

In this paper we described a framework for solving multiclass problems via pairs of embeddings. The proposed framework can be viewed as a generalization of ECOC with logistic regressors. It is possible to extend our framework in a few ways. First, the probabilistic embeddings can be replaced with non-negative embeddings by replacing the logistic transformation with the exponential function. In this case, the KL divergence is replaced with its unormalized version [2, 9]. The resulting generalized Bunching algorithm is somewhat more involved and less intuitive to understand. Second, while our work focuses on linear embeddings, our algorithm and analysis can be adapted to more complex mappings by employing *kernel operators*. This can be achieved by replacing the $k$'th scalar-product $T_k \cdot x$ with an abstract inner-product $\kappa(T_k, x)$. Last, we would like to note that it is possible to devise an alternative objective function to the one given in Eq. (9) which is jointly convex in $(T, \sigma(C))$ and for which we can state a bound of a form similar to the bound in Thm. 1.

## References

[1] E.L. Allwein, R.E. Schapire, and Y. Singer. Reducing multiclass to binary: A unifying approach for margin classifiers. *Journal of Machine Learning Research*, 1:113–141, 2000.

[2] Michael Collins, Robert E. Schapire, and Yoram Singer. Logistic regression, adaboost and bregman distances. *Machine Learning*, 47(2/3):253–285, 2002.

[3] K. Crammer and Y. Singer. On the learnability and design of output codes for multiclass problems. In *Proc. of the Thirteenth Annual Conference on Computational Learning Theory*, 2000.

[4] I. Csiszár and G. Tusnády. Information geometry and alternaning minimization procedures. *Statistics and Decisions, Supplement Issue*, 1:205–237, 1984.

[5] A.P. Dempster, N.M. Laird, and D.B. Rubin. Maximum likelihood from incomplete data via the EM algorithm. *Journal of the Royal Statistical Society, Ser. B*, 39:1–38, 1977.

[6] Thomas G. Dietterich and Ghulum Bakiri. Solving multiclass learning problems via error-correcting output codes. *Journal of Artificial Intelligence Research*, 2:263–286, January 1995.

[7] Jyrki Kivinen and Manfred K. Warmuth. Additive versus exponentiated gradient updates for linear prediction. *Information and Computation*, 132(1):1–64, January 1997.

[8] John D. Lafferty. Additive models, boosting and inference for generalized divergences. In *Proceedings of the Twelfth Annual Conference on Computational Learning Theory*, 1999.

[9] S. Della Pietra, V. Della Pietra, and J. Lafferty. Duality and auxilary functions for Bregman distances. Technical Report CS-01-10, CMU, 2002.

[10] T. Poggio and F. Girosi. Networks for approximation and learning. *Proc. of IEEE*, 78(9), 1990.

[11] R.E. Schapire, M. Rochery, M. Rahim, and N. Gupta. Incorporating prior knowledge into boosting. In *Machine Learning: Proceedings of the Nineteenth International Conference*, 2002.
